# Minimax Localization of Structural Information in Large Noisy Matrices

**Mladen Kolar**[†*]          **Sivaraman Balakrishnan**[†*]          **Alessandro Rinaldo**[††]
mladenk@cs.cmu.edu      sbalakri@cs.cmu.edu       arinaldo@stat.cmu.edu

**Aarti Singh**[†]
aarti@cs.cmu.edu

[†] **School of Computer Science and** [††] **Department of Statistics, Carnegie Mellon University**

## Abstract

We consider the problem of identifying a sparse set of relevant columns and rows in a large data matrix with highly corrupted entries. This problem of identifying groups from a collection of bipartite variables such as proteins and drugs, biological species and gene sequences, malware and signatures, etc is commonly referred to as biclustering or co-clustering. Despite its great practical relevance, and although several ad-hoc methods are available for biclustering, theoretical analysis of the problem is largely non-existent. The problem we consider is also closely related to structured multiple hypothesis testing, an area of statistics that has recently witnessed a flurry of activity. We make the following contributions

1. We prove lower bounds on the minimum signal strength needed for successful recovery of a bicluster as a function of the noise variance, size of the matrix and bicluster of interest.
2. We show that a combinatorial procedure based on the scan statistic achieves this optimal limit.
3. We characterize the SNR required by several computationally tractable procedures for biclustering including element-wise thresholding, column/row average thresholding and a convex relaxation approach to sparse singular vector decomposition.

## 1   Introduction

Biclustering is the problem of identifying a (typically) sparse set of relevant columns and rows in a large, noisy data matrix. This problem along with the first algorithm to solve it were proposed by Hartigan [14] as a way to directly cluster data matrices to produce clusters with greater interpretability. Biclustering routinely arises in several applications such as discovering groups of proteins and drugs that interact with each other [19], learning phylogenetic relationships between different species based on alignments of snippets of their gene sequences [30], identifying malware that have similar signatures [7] and identifying groups of users with similar tastes for commercial products [29]. In these applications, the data matrix is often indexed by (object, feature) pairs and the goal is to identify clusters in this set of bipartite variables.

In standard clustering problems, the goal is only to identify meaningful groups of objects and the methods typically use the entire feature vector to define a notion of similarity between the objects.

---

[*]These authors contributed equally to this work

Biclustering can be thought of as high-dimensional clustering where only a subset of the features are relevant for identifying similar objects, and the goal is to identify not only groups of objects that are similar, but also which features are relevant to the clustering task. Consider, for instance gene expression data where the objects correspond to genes, and the features correspond to their expression levels under a variety of experimental conditions. Our present understanding of biological systems leads us to expect that subsets of genes will be co-expressed only under a small number of experimental conditions. Although, pairs of genes are not expected to be similar under *all* experimental conditions it is critical to be able to discover local expression patterns, which can for instance correspond to joint participation in a particular biological pathway or process. Thus, while clustering aims to identify *global* structure in the data, biclustering take a more *local* approach by jointly clustering *both* objects and features.

Prevalent techniques for finding biclusters are typically heuristic procedures with little or no theoretical underpinning. In order to study, understand and compare biclustering algorithms we consider a simple theoretical model of biclustering [18, 17, 26]. This model is akin to the spiked covariance model of [15] widely used in the study of PCA in high-dimensions.

We will focus on the following simple observation model for the matrix $\mathbf{A} \in \mathbb{R}^{n_1 \times n_2}$:

$$\mathbf{A} = \beta \mathbf{u}\mathbf{v}' + \mathbf{\Delta} \tag{1}$$

where $\mathbf{\Delta} = \{\Delta_{ij}\}_{i \in [n_1], j \in [n_2]}$ is a random matrix whose entries are i.i.d. $\mathcal{N}(0, \sigma^2)$ with $\sigma^2 > 0$ known, $\mathbf{u} = \{u_i : i \in [n_1]\}$ and $\mathbf{v} = \{v_i : i \in [n_2]\}$ are unknown deterministic unit vectors in $\mathbb{R}^{n_1}$ and $\mathbb{R}^{n_2}$, respectively, and $\beta > 0$ is a constant. To simplify the presentation, we assume that $\mathbf{u} \propto \{0,1\}^{n_1}$ and $\mathbf{v} \propto \{0,1\}^{n_2}$. Let $K_1 = \{i \ : \ u_i \neq 0\}$ and $K_2 = \{i \ : \ v_i \neq 0\}$ be the sets indexing the non-zero components of the vectors $\mathbf{u}$ and $\mathbf{v}$, respectively. We assume that $\mathbf{u}$ and $\mathbf{v}$ are sparse, that is, $k_1 := |K_1| \ll n_1$ and $k_2 := |K_2| \ll n_2$. While the sets $(K_1, K_2)$ are unknown, we assume that their cardinalities are known. Notice that the magnitude of the signal for all the coordinates in the bicluster $K_1 \times K_2$ is $\frac{\beta}{\sqrt{k_1 k_2}}$. The parameter $\beta$ measures the strength of the signal, and is the key quantity we will be studying.

We focus on the case of a single bicluster that appears as an elevated sub-matrix of size $k_1 \times k_2$ with signal strength $\beta$ embedded in a large $n_1 \times n_2$ data matrix with entries corrupted by additive Gaussian noise with variance $\sigma^2$. Under this model, the biclustering problem is formulated as the problem of estimating the sets $K_1$ and $K_2$, based on a single noisy observation $\mathbf{A}$ of the unknown signal matrix $\beta \mathbf{u}\mathbf{v}'$. Biclustering is most subtle when the matrix is large with several irrelevant variables, the entries are highly noisy, and the bicluster is small as defined by a sparse set of rows/columns. We provide a sharp characterization of tuples of $(\beta, n_1, n_2, k_1, k_2, \sigma^2)$ under which it is possible to recover the bicluster and study several common methods and establish the regimes under which they succeed.

We establish minimax lower and upper bounds for the following class of models. Let

$$\Theta(\beta_0, k_1, k_2) := \{(\beta, K_1, K_2) \ : \ \beta \geq \beta_0, |K_1| = k_1, K_1 \subset [n_1], |K_2| = k_2, K_2 \subset [n_2]\} \tag{2}$$

be a set of parameters. For a parameter $\theta \in \Theta$, let $\mathbb{P}_\theta$ denote the joint distribution of the entries of $\mathbf{A} = \{a_{ij}\}_{i \in [n_1], j \in [n_2]}$, whose density with respect to the Lebesgue measure is

$$\prod_{ij} \mathcal{N}(a_{ij}; \beta(k_1 k_2)^{-1/2} \, \mathbb{1}\{i \in K_1, j \in K_2\}, \sigma^2), \tag{3}$$

where the notation $\mathcal{N}(z; \mu, \sigma^2)$ denotes the distribution $p(z) \sim \mathcal{N}(\mu, \sigma^2)$ of a Gaussian random variable with mean $\mu$ and variance $\sigma^2$, and $\mathbb{1}$ denotes the indicator function.

We derive a lower bound that identifies tuples of $(\beta, n_1, n_2, k_1, k_2, \sigma^2)$ under which we can recover the true biclustering from a noisy high dimensional matrix. We show that a combinatorial procedure based on the scan statistic achieves the minimax optimal limits, however it is impractical as it requires enumerating all possible sub-matrices of a given size in a large matrix. We analyze the scalings (i.e. the relation between $\beta$ and $(n_1, n_2, k_1, k_2, \sigma^2)$) under which some computationally tractable procedures for biclustering including element-wise thresholding, column/row average thresholding and sparse singular vector decomposition (SSVD) succeed with high probability.

We consider the detection of both small and large biclusters of weak activation, and show that at the minimax scaling the problem is surprisingly subtle (e.g., even detecting big clusters is quite hard).

In Table 1, we describe our main findings and compare the scalings under which the various algorithms succeed.

| Algorithm | Combinatorial | Thresholding | Row/Column Averaging | Sparse SVD |
|---|---|---|---|---|
| SNR scaling | Minimax | Weak | Intermediate | Weak |
| Bicluster size | Any | Any | $(n_1^{1/2+\alpha} \times n_2^{1/2+\alpha}), \alpha \in (0, 1/2)$ | Any |
| | Theorem 2 | Theorem 3 | Theorem 4 | Theorem 5 |

Where the scalings are,

1. Minimax: $\beta \sim \sigma \max\left(\sqrt{k_1 \log(n_1 - k_1)}, \sqrt{k_2 \log(n_2 - k_2)}\right)$
2. Weak: $\beta \sim \sigma \max\left(\sqrt{k_1 k_2 \log(n_1 - k_1)}, \sqrt{k_1 k_2 \log(n_2 - k_2)}\right)$
3. Intermediate (for large clusters): $\beta \sim \sigma \max\left(\frac{\sqrt{k_1 k_2 \log(n_1 - k_1)}}{n_2^\alpha}, \frac{\sqrt{k_1 k_2 \log(n_2 - k_2)}}{n_1^\alpha}\right)$

Element-wise thresholding does not take advantage of any structure in the data matrix and hence does not achieve the minimax scaling for any bicluster size. If the clusters are big enough row/column averaging performs better than element-wise thresholding since it can take advantage of structure. We also study a convex relaxation for sparse SVD, based on the DSPCA algorithm proposed by [11] that encourages the singular vectors of the matrix to be supported over a sparse set of variables. However, despite the increasing popularity of this method, we show that it is only guaranteed to yield a sparse set of singular vectors when the SNR is quite high, equivalent to element-wise thresholding, and fails for stronger scalings of the SNR.

## 1.1 Related work

Due to its practical importance and difficulty biclustering has attracted considerable attention (for some recent surveys see [9, 27, 20, 22]). Broadly algorithms for biclustering can be categorized as either score-based searches, or spectral algorithms. Many of the proposed algorithms for identifying relevant clusters are based on heuristic searches whose goal is to identify large average sub-matrices or sub-matrices that are well fit by a two-way ANOVA model. Sun et. al. [26] provide some statistical backing for these exhaustive search procedures. In particular, they show how to construct a test via exhaustive search to distinguish when there is a small sub-matrix of weak activation from the "null" case when there is no bicluster.

The premise behind the spectral algorithms is that if there was a sub-matrix embedded in a large matrix, then this sub-matrix could be identified from the left and right singular vectors of $\mathbf{A}$. In the case when exactly one of $\mathbf{u}$ and $\mathbf{v}$ is random, the model (1) can be related to the spiked covariance model of [15]. In the case when $\mathbf{v}$ is random, the matrix $\mathbf{A}$ has independent columns and dependent rows. Therefore, $\mathbf{A}'\mathbf{A}$ is a spiked covariance matrix and it is possible to use the existing theoretical results on the first eigenvalue to characterize the left singular vector of $\mathbf{A}$. A lot of recent work has dealt with estimation of sparse eigenvectors of $\mathbf{A}'\mathbf{A}$, see for example [32, 16, 24, 31, 2]. For biclustering applications, the assumption that exactly one $\mathbf{u}$ or $\mathbf{v}$ is random, is not justifiable, therefore, theoretical results for the spiked covariance model do not translate directly. Singular vectors of the model (1) have been analyzed by [21], improving on earlier results of [6]. These results however are asymptotic and do not consider the case when $\mathbf{u}$ and $\mathbf{v}$ are sparse.

Our setup for the biclustering problem also falls in the framework of structured normal means multiple hypothesis testing problems, where for each entry in the matrix the hypotheses are that the entry has mean 0 versus an elevated mean. The presence of a bicluster (sub-matrix) however imposes structure on which elements are elevated concurrently. Recently, several papers have investigated the structured normal means setting for ordered domains. For example, [5] consider the detection of elevated intervals and other parametric structures along an ordered line or grid, [4] consider detection of elevated connected paths in tree and lattice topologies, [3] considers nonparametric cluster structures in a regular grid. In addition, [1] consider testing of different elevated structures in a general but known graph topology. Our setup for the biclustering problem requires identification of an elevated submatrix in an *unordered* matrix. At a high level, all these results suggest that it is possible to leverage the structure to improve the SNR threshold at which the hypothesis testing problem is

feasible. However, computationally efficient procedures that achieve the minimax SNR thresholds are only known for a few of these problems. Our results for biclustering have a similar flavor, in that the minimax threshold requires a combinatorial procedure whereas the computationally efficient procedures we investigate are often sub-optimal.

The rest of this paper is organized as follows. In Section 2, we provide a lower bound on the minimum signal strength needed for successfully identifying the bicluster. Section 3 presents a combinatorial procedure which achieves the lower bound and hence is minimax optimal. We investigate some computationally efficient procedures in Section 4. Simulation results are presented in Section 5 and we conclude in Section 6. All proofs are deferred to the Appendix.

## 2   Lower bound

In this section, we derive a lower bound for the problem of identifying the correct bicluster, indexed by $K_1$ and $K_2$, in model (1). In particular, we derive conditions on $(\beta, n_1, n_2, k_1, k_2, \sigma^2)$ under which any method is going to make an error when estimating the correct cluster. Intuitively, if either the signal-to-noise ratio $\beta/\sigma$ or the cluster size is small, the minimum signal strength needed will be high since it is harder to distinguish the bicluster from the noise.

**Theorem 1.** *Let $\alpha \in (0, \frac{1}{8})$ and*

$$
\begin{aligned}
\beta_{\min} &= \beta_{\min}(n_1, n_2, k_1, k_2, \sigma) \\
&= \sigma\sqrt{\alpha}\max\left(\sqrt{k_1 \log(n_1 - k_1)}, \sqrt{k_2 \log(n_2 - k_2)}, \sqrt{\frac{k_1 k_2 \log(n_1 - k_1)(n_2 - k_1)}{k_1 + k_2 - 1}}\right).
\end{aligned}
\tag{4}
$$

*Then for all $\beta_0 \leq \beta_{\min}$,*

$$
\inf_{\Phi}\ \sup_{\theta \in \Theta(\beta_0, k_1, k_2)} \mathbb{P}_\theta[\Phi(\mathbf{A}) \neq (K_1(\theta), K_2(\theta))] \geq \frac{\sqrt{M}}{1 + \sqrt{M}}\left(1 - 2\alpha - \frac{2\alpha}{\log M}\right) \xrightarrow{n_1, n_2 \to \infty} 1 - 2\alpha,
\tag{5}
$$

*where $M = \min(n_1 - k_1, n_2 - k_2)$, $\Theta(\beta_0, k_1, k_2)$ is given in (2) and the infimum is over all measurable maps $\Phi : \mathbb{R}^{n_1 \times n_2} \mapsto 2^{[n_1]} \times 2^{[n_2]}$.*

The result can be interpreted in the following way: for any biclustering procedure $\Phi$, if $\beta_0 \leq \beta_{\min}$, then there exists some element in the model class $\Theta(\beta_0, k_1, k_2)$ such that the probability of incorrectly identifying the sets $K_1$ and $K_2$ is bounded away from zero.

The proof is based on a standard technique described in Chapter 2.6 of [28]. We start by identifying a subset of parameter tuples that are hard to distinguish. Once a suitable finite set is identified, tools for establishing lower bounds on the error in multiple-hypothesis testing can be directly applied. These tools only require computing the Kullback-Leibler (KL) divergence between two distributions $\mathbb{P}_{\theta_1}$ and $\mathbb{P}_{\theta_2}$, which in the case of model (1) are two multivariate normal distributions. These constructions and calculations are described in detail in the Appendix.

## 3   Minimax optimal combinatorial procedure

We now investigate a combinatorial procedure achieving the lower bound of Theorem 1, in the sense that, for any $\theta \in \Theta(\beta_{\min}, k_1, k_2)$, the probability of recovering the true bicluster $(K_1, K_2)$ tends to one as $n_1$ and $n_2$ grow unbounded. This scan procedure consists in enumerating all possible pairs of subsets of the row and column indexes of size $k_1$ and $k_2$, respectively, and choosing the one whose corresponding submatrix has the largest overall sum. In detail, for an observed matrix $\mathbf{A}$ and two candidate subsets $\tilde{K}_1 \subset [n_1]$ and $\tilde{K}_2 \subset [n_2]$, we define the associated score $\mathcal{S}(\tilde{K}_1, \tilde{K}_2) := \sum_{i \in \tilde{K}_1} \sum_{j \in \tilde{K}_2} a_{ij}$. The estimated bicluster is the pair of subsets of sizes $k_1$ and $k_2$ achieving the highest score:

$$
\Psi(\mathbf{A}) := \operatorname*{argmax}_{(\tilde{K}_1, \tilde{K}_2)} \mathcal{S}(\tilde{K}_1, \tilde{K}_2) \quad \text{subject to} \quad |\tilde{K}_1| = k_1, |\tilde{K}_2| = k_2.
\tag{6}
$$

The following theorem determines the signal strength $\beta$ needed for the decoder $\Psi$ to find the true bicluster.

**Theorem 2.** *Let* $\mathbf{A} \sim \mathbb{P}_\theta$ *with* $\theta \in \Theta(\beta, k_1, k_2)$ *and assume that* $k_1 \leq n_1/2$ *and* $k_2 \leq n_2/2$. *If*

$$\beta \geq 4\sigma \max \left( \sqrt{k_1 \log(n_1 - k_1)}, \sqrt{k_2 \log(n_2 - k_2)}, \sqrt{\frac{2k_1 k_2 \log(n_1 - k_1)(n_2 - k_2)}{k_1 + k_2}} \right) \quad (7)$$

*then* $\mathbb{P}[\Psi(\mathbf{A}) \neq (K_1, K_2)] \leq 2[(n_1 - k_1)^{-1} + (n_2 - k_2)^{-1}]$ *where* $\Psi$ *is the decoder defined in* (6).

Comparing to the lower bound in Theorem 1, we observe that the combinatorial procedure using the decoder $\Psi$ that looks for all possible clusters and chooses the one with largest score achieves the lower bound up to constants. Unfortunately, this procedure is not practical for data sets commonly encountered in practice, as it requires enumerating all $\binom{n_1}{k_1} \binom{n_2}{k_2}$ possible sub-matrices of size $k_1 \times k_2$. The combinatorial procedure requires the signal to be positive, but not necessarily constant throughout the bicluster. In fact it is easy to see that provided the average signal in the bicluster is larger than that stipulated by the theorem this procedure succeeds with high probability irrespective of how the signal is distributed across the bicluster. Finally, we remark that the estimation of the cluster is done under the assumption that $k_1$ and $k_2$ are known. Establishing minimax lower bounds and a procedure that adapts to unknown $k_1$ and $k_2$ is an open problem.

## 4 Computationally efficient biclustering procedures

In this section we investigate the performance of various procedures for biclustering, that, unlike the optimal scan statistic procedure studied in the previous section, are computationally tractable. For each of these procedures however, computational ease comes at the cost of suboptimal performance: recovery of the true bicluster is only possible if the $\beta$ is much larger than the minimax signal strength of Theorem 1.

### 4.1 Element-wise thresholding

The simplest procedure that we analyze is based on element-wise thresholding. The bicluster is estimated as

$$\Psi_{\mathrm{thr}}(\mathbf{A}, \tau) := \{(i, j) \in [n_1] \times [n_2] : |a_{ij}| \geq \tau\} \quad (8)$$

where $\tau > 0$ is a parameter. The following theorem characterizes the signal strength $\beta$ required for the element-wise thresholding to succeed in recovering the bicluster.

**Theorem 3.** *Let* $\mathbf{A} \sim \mathbb{P}_\theta$ *with* $\theta \in \Theta(\beta, k_1, k_2)$ *and fix* $\delta > 0$. *Set the threshold* $\tau$ *as*

$$\tau = \sigma \sqrt{2 \log \frac{(n_1 - k_1)(n_2 - k_2) + k_1(n_2 - k_2) + k_2(n_1 - k_1)}{\delta}}.$$

*If*

$$\beta \geq \sqrt{k_1 k_2} \sigma \left( \sqrt{2 \log \frac{k_1 k_2}{\delta}} + \sqrt{2 \log \frac{(n_1 - k_1)(n_2 - k_2) + k_1(n_2 - k_2) + k_2(n_1 - k_1)}{\delta}} \right)$$

*then* $\mathbb{P}[\Psi_{\mathrm{thr}}(\mathbf{A}, \tau) \neq K_1 \times K_2] = o(\delta/(k_1 k_2))$.

Comparing Theorem 3 with the lower bound in Theorem 1, we observe that the signal strength $\beta$ needs to be $\mathcal{O}(\max(\sqrt{k_1}, \sqrt{k_2}))$ larger than the lower bound. This is not surprising, since the element-wise thresholding is not exploiting the structure of the problem, but is assuming that the large elements of the matrix $\mathbf{A}$ are positioned randomly. From the proof it is not hard to see that this upper bound is tight up to constants, i.e. if $\beta \leq c\sqrt{k_1 k_2} \sigma \left( \sqrt{2 \log \frac{k_1 k_2}{\delta}} + \sqrt{2 \log \frac{(n_1-k_1)(n_2-k_2)+k_1(n_2-k_2)+k_2(n_1-k_1)}{\delta}} \right)$ for a small enough constant $c$ then thresholding will no longer recover the bicluster with probability at least $1 - \delta$. It is also worth noting that thresholding neither requires the signal in the bicluster to be constant nor positive provided it is larger in magnitude, at every entry, than the threshold specified in the theorem.

## 4.2 Row/Column averaging

Next, we analyze another a procedure based on column and row averaging. When the bicluster is large this procedure exploits the structure of the problem and outperforms the simple element-wise thresholding and the sparse SVD, which is discussed in the following section. The averaging procedure works only well if the bicluster is "large", as specified below, since otherwise the row or column average is dominated by the noise.

More precisely, the averaging procedure computes the average of each row and column of $\mathbf{A}$ and outputs the $k_1$ rows and $k_2$ columns with the largest average. Let $\{r_{r,i}\}_{i\in[n_1]}$ and $\{r_{c,j}\}_{j\in[n_2]}$ denote the positions of rows and columns when they are ordered according to row and column averages in descending order. The bicluster is estimated then as

$$\Psi_{\mathrm{avg}}(\mathbf{A}) := \{i \in [n_1] : r_{r,i} \leq k_1\} \times \{j \in [n_2] : r_{c,j} \leq k_2\}. \tag{9}$$

The following theorem characterizes the signal strength $\beta$ required for the averaging procedure to succeed in recovering the bicluster.

**Theorem 4.** *Let $\mathbf{A} \sim \mathbb{P}_\theta$ with $\theta \in \Theta(\beta, k_1, k_2)$. If $k_1 = \Omega(n_1^{1/2+\alpha})$ and $k_2 = \Omega(n_2^{1/2+\alpha})$, where $\alpha \in (0, 1/2)$ is a constant and,*

$$\beta \geq 4\sigma \max\left( \frac{\sqrt{k_1 k_2 \log(n_1 - k_1)}}{n_2^\alpha}, \frac{\sqrt{k_1 k_2 \log(n_2 - k_2)}}{n_1^\alpha} \right)$$

*then $\mathbb{P}[\Psi(\mathbf{A}) \neq (K_1, K_2)] \leq [n_1^{-1} + n_2^{-1}]$.*

Comparing to Theorem 3, we observe that the averaging requires lower signal strength than the element-wise thresholding when the bicluster is large, that is, $k_1 = \Omega(\sqrt{n_1})$ and $k_2 = \Omega(\sqrt{n_2})$. Unless both $k_1 = \mathcal{O}(n_1)$ and $k_2 = \mathcal{O}(n_2)$, the procedure does not achieve the lower bound of Theorem 1, however, the procedure is simple and computationally efficient. It is also not hard to show that this theorem is sharp in its characterization of the averaging procedure. Further, unlike thresholding, averaging requires the signal to be positive in the bicluster.

It is interesting to note that a large bicluster can also be identified without assuming the normality of the noise matrix $\mathbf{\Delta}$. This non-parametric extension is based on a simple sign-test, and the details are provided in Appendix.

## 4.3 Sparse singular value decomposition (SSVD)

An alternate way to estimate $K_1$ and $K_2$ would be based on the singular value decomposition (SVD), i.e. finding $\tilde{\mathbf{u}}$ and $\tilde{\mathbf{v}}$ that maximize $\langle \tilde{\mathbf{u}}, \mathbf{A}\tilde{\mathbf{v}} \rangle$, and then threshold the elements of $\tilde{\mathbf{u}}$ and $\tilde{\mathbf{v}}$. Unfortunately, such a method would perform poorly when the signal $\beta$ is weak and the dimensionality is high, since, due to the accumulation of noise, $\tilde{\mathbf{u}}$ and $\tilde{\mathbf{v}}$ are poor estimates of $\mathbf{u}$ and $\mathbf{v}$ and and do not exploit the fact that $\mathbf{u}$ and $\mathbf{v}$ are sparse.

In fact, it is now well understood [8] that SVD is strongly inconsistent when the signal strength is weak, i.e. $\angle(\tilde{\mathbf{u}}, \mathbf{u}) \to \pi/2$ (and similarly for $\mathbf{v}$) almost surely. See [26] for a clear exposition and discussion of this inconsistency in the SVD setting.

To properly exploit the sparsity in the singular vectors, it seems natural to impose a cardinality constraint to obtain a sparse singular vector decomposition (SSVD):

$$\max_{\mathbf{u}\in\mathbf{S}^{n_1-1}, \mathbf{v}\in\mathbf{S}^{n_2-1}} \langle \mathbf{u}, \mathbf{A}\mathbf{v} \rangle \quad \text{subject to} \quad ||\mathbf{u}||_0 \leq k_1, \ ||\mathbf{v}||_0 \leq k_2,$$

which can be further rewritten as

$$\max_{\mathbf{Z}\in\mathbb{R}^{n_2 \times n_1}} \operatorname{tr} \mathbf{A}\mathbf{Z} \quad \text{subject to} \quad \mathbf{Z} = \mathbf{v}\mathbf{u}', \ ||\mathbf{u}||_2 = 1, \ ||\mathbf{v}||_2 = 1, \ ||\mathbf{u}||_0 \leq k_1, \ ||\mathbf{v}||_0 \leq k_2. \tag{10}$$

The above problem is non-convex and computationally intractable.

Inspired by the convex relaxation methods for sparse principal component analysis proposed by [11], we consider the following relaxation the SSVD:

$$\max_{\mathbf{X}\in\mathbb{R}^{(n_1+n_2)\times(n_1+n_2)}} \operatorname{tr} \mathbf{A}\mathbf{X}^{21} - \lambda \mathbf{1}'|\mathbf{X}^{21}|\mathbf{1} \quad \text{subject to} \quad \mathbf{X} \succeq \mathbf{0}, \ \operatorname{tr} \mathbf{X}^{11} = 1, \operatorname{tr} \mathbf{X}^{22} = 1, \tag{11}$$

where $\mathbf{X}$ is the block matrix

$$\left[ \begin{array}{cc} \mathbf{X}^{11} & \mathbf{X}^{12} \\ \mathbf{X}^{21} & \mathbf{X}^{22} \end{array} \right]$$

with the block $\mathbf{X}^{21}$ corresponding to $\mathbf{Z}$ in (10). If the optimal solution $\widehat{\mathbf{X}}$ is of rank 1, then, necessarily, $\widehat{\mathbf{X}} = \left( \begin{smallmatrix} \widehat{\mathbf{u}} \\ \widehat{\mathbf{v}} \end{smallmatrix} \right) (\widehat{\mathbf{u}}' \ \widehat{\mathbf{v}}')$. Based on the sparse singular vectors $\widehat{\mathbf{u}}$ and $\widehat{\mathbf{v}}$, we estimate the bicluster as

$$\widehat{K}_1 = \{ j \in [n_1] \ : \ \widehat{u}_j \neq 0 \} \qquad \text{and} \qquad \widehat{K}_2 = \{ j \in [n_2] \ : \ \widehat{v}_j \neq 0 \}. \tag{12}$$

The user defined parameter $\lambda$ controls the sparsity of the solution $\widehat{\mathbf{X}}^{21}$, and, therefore, provided the solution is of rank one, it also controls the sparsity of the vectors $\widehat{\mathbf{u}}$ and $\widehat{\mathbf{v}}$ and of the estimated bicluster.

The following theorem provides *sufficient* conditions for the solution $\widehat{\mathbf{X}}$ to be rank one and to recover the bicluster.

**Theorem 5.** *Consider the model in* (1). *Assume $k_1 \asymp k_2$ and $k_1 \leq n_1/2$ and $k_2 \leq n_2/2$. If*

$$\beta \geq 2\sigma \sqrt{k_1 k_2 \log(n_1 - k_1)(n_2 - k_2)} \tag{13}$$

*then the solution $\widehat{\mathbf{X}}$ of the optimization problem in* (11) *with $\lambda = \frac{\beta}{2\sqrt{k_1 k_2}}$ is of rank 1 with probability $1 - \mathcal{O}(k_1^{-1})$. Furthermore, we have that $(\widehat{K}_1, \widehat{K}_2) = (K_1, K_2)$ with probability $1 - \mathcal{O}(k_1^{-1})$.*

It is worth noting that SSVD correctly recovers *signed* vectors $\widehat{u}$ and $\widehat{v}$ under this signal strength. In particular, the procedure works even if the $u$ and $v$ in Equation 1 are signed.

The following theorem establishes *necessary* conditions for the SSVD to have a rank 1 solution that correctly identifies the bicluster.

**Theorem 6.** *Consider the model in* (1). *Fix $c \in (0, 1/2)$. Assume that $k_1 \asymp k_2$ and $k_1 = o(n^{1/2-c})$ and $k_2 = o(n_2^{1/2-c})$. If*

$$\beta \leq 2\sigma \sqrt{ck_1 k_2 \log \max(n_1 - k_1, n_2 - k_2)}, \tag{14}$$

*with $\lambda = \frac{\beta}{2\sqrt{k_1 k_2}}$ then the optimization problem* (11) *does not have a rank 1 solution that correctly recovers the sparsity pattern with probability at least $1 - \mathcal{O}(\exp(-(\sqrt{k_1} + \sqrt{k_2})^2))$ for sufficiently large $n_1$ and $n_2$.*

From Theorem 6 observe that the sufficient conditions of Theorem 5 are sharp. In particular, the two theorems establish that the SSVD does not establish the lower bound given in Theorem 1. The signal strength needs to be of the same order as for the element-wise thresholding, which is somewhat surprising since from the formulation of the SSVD optimization problem it seems that the procedure uses the structure of the problem. From numerical simulations in Section 5 we observe that although SSVD requires the same scaling as thresholding, it consistently performs slightly better at a fixed signal strength.

## 5 Simulation results

We test the performance of the three computationally efficient procedures on synthetic data: thresholding, averaging and sparse SVD. For sparse SVD we use an implementation posted online by [11]. We generate data from (1) with $n = n_1 = n_2$, $k = k_1 = k_2$, $\sigma^2 = 1$ and $\mathbf{u} = \mathbf{v} \propto (\mathbf{1}'_k, \mathbf{0}'_{n-k})'$. For each algorithm we plot the Hamming fraction (i.e. the Hamming distance between $\mathbf{s}_{\widehat{u}}$ and $\mathbf{s}_u$ rescaled to be between 0 and 1) against the rescaled sample size. In each case we average the results over 50 runs.

For thresholding and sparse SVD the rescaled scaling (x-axis) is $\frac{\beta}{k\sqrt{\log(n-k)}}$ and for averaging the rescaled scaling (x-axis) is $\frac{\beta n^{\alpha}}{k\sqrt{\log(n-k)}}$. We observe that there is a sharp threshold between success and failure of the algorithms, and the curves show good agreement with our theory.

The vertical line shows the point after which successful recovery happens for all values of $n$. We can make a direct comparison between thresholding and sparse SVD (since the curves are identically rescaled) to see that at least empirically sparse SVD succeeds at a smaller scaling constant than thresholding even though their asymptotic rates are identical.

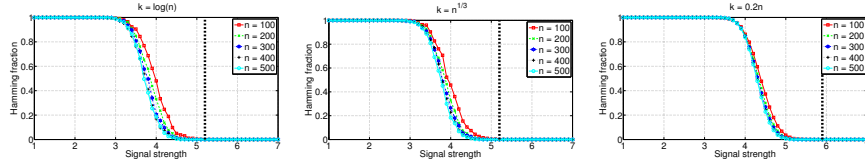

Figure 1: Thresholding: Hamming fraction versus rescaled signal strength.

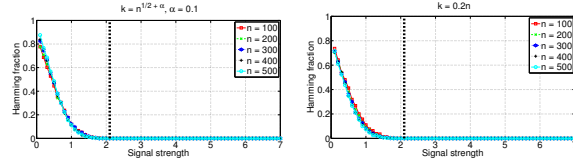

Figure 2: Averaging: Hamming fraction versus rescaled signal strength.

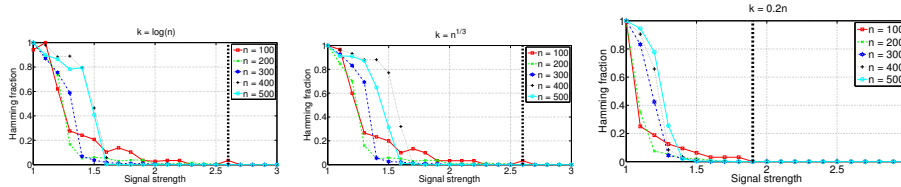

Figure 3: Sparse SVD: Hamming fraction versus rescaled signal strength.

## 6 Discussion

In this paper, we analyze biclustering using a simple statistical model (1), where a sparse rank one matrix is perturbed with noise. Using this model, we have characterized the minimal signal strength below which no procedure can succeed in recovering the bicluster. This lower bound can be matched using an exhaustive search technique. However, it is still an open problem to find a computationally efficient procedure that is minimax optimal.

Amini et. al. [2] analyze the convex relaxation procedure proposed in [11] for high-dimensional sparse PCA. Under the minimax scaling for this problem they show that provided a rank-1 solution exists it has the desired sparsity pattern (they were however not able to show that a rank-1 solution exists with high probability). Somewhat surprisingly, we show that in the SVD case a rank-1 solution with the desired sparsity pattern *does not* exist with high probability. The two settings however are not identical since the noise in the spiked covariance model is Wishart rather than Gaussian, and has correlated entries. It would be interesting to analyze whether our negative result has similar implications for the sparse PCA setting.

The focus of our paper has been on a model with one cluster, which although simple, provides several interesting theoretical insights. In practice, data often contains multiple clusters which need to be estimated. Many existing algorithms (see e.g. [17] and [18]) try to estimate multiple clusters and it would be useful to analyze these theoretically.

Furthermore, the algorithms that we have analyzed assume knowledge of the size of the cluster, which is used to select the tuning parameters. It is a challenging problem of great practical relevance to find data driven methods to select these tuning parameters.

## 7 Acknowledgments

We would like to thank Arash Amini and Martin Wainwright for fruitful discussions, and Larry Wasserman for his ideas, indispensable advice and wise guidance. This research is supported in part by AFOSR under grant FA9550-10-1-0382 and NSF under grant IIS-1116458. SB would also like to thank Jaime Carbonell and Srivatsan Narayanan for several valuable comments and thought-provoking discussions.

# References

[1] Louigi Addario-Berry, Nicolas Broutin, Luc Devroye, and Gábor Lugosi. On combinatorial testing problems. *Ann. Statist.*, 38(5):3063–3092, 2010.

[2] A.A. Amini and M.J. Wainright. High-Dimensional Analysis Of Semidefinite Relaxations For Sparse Principal Components. *The Annals of Statistics*, 37(5B):2877–2921, 2009.

[3] Ery Arias-Castro, Emmanuel J. Candès, and Arnaud Durand. Detection of an anomalous cluster in a network. *Ann. Stat.*, 39(1):278–304, 2011.

[4] Ery Arias-Castro, Emmanuel J. Candès, Hannes Helgason, and Ofer Zeitouni. Searching for a trail of evidence in a maze. *Ann. Statist.*, 36(4):1726–1757, 2008.

[5] Ery Arias-Castro, David L. Donoho, and Xiaoming Huo. Adaptive multiscale detection of filamentary structures in a background of uniform random points. *Ann. Statist.*, 34(1):326–349, 2006.

[6] Jushan Bai. Inferential theory for factor models of large dimensions. *Econometrica*, 71(1):pp. 135–171, 2003.

[7] Ulrich Bayer, Paolo Milani Comparetti, Clemens Hlauscheck, Christopher Kruegel, and Engin Kirda. Scalable, Behavior-Based Malware Clustering. In *16th Symposium on Network and Distributed System Security (NDSS)*, 2009.

[8] F. Benaych-Georges and R. Rao Nadakuditi. The singular values and vectors of low rank perturbations of large rectangular random matrices. *ArXiv e-prints*, March 2011.

[9] S. Busygin, O. Prokopyev, and P.M. Pardalos. Biclustering in data mining. *Computers & Operations Research*, 35(9):2964–2987, 2008.

[10] Emmanuel J. Candès, Xiaodong Li, Yi Ma, and John Wright. Robust principal component analysis? *CoRR*, abs/0912.3599, 2009.

[11] Alexandre d'Aspremont, Laurent El Ghaoui, Michael I. Jordan, and Gert R. G. Lanckriet. A direct formulation for sparse pca using semidefinite programming. *SIAM Review*, 49:434–448, 2007.

[12] K.R. Davidson and S.J. Szarek. Local operator theory, random matrices and Banach spaces. *Handbook of the geometry of Banach spaces*, 1:317–366, 2001.

[13] R. Fletcher. Semi-definite matrix constraints in optimization. *SIAM Journal on Control and Optimization*, 23:493, 1985.

[14] J. A. Hartigan. Direct clustering of a data matrix. *Journal of the American Statistical Association*, 67(337):pp. 123–129, 1972.

[15] I.M. Johnstone. On the distribution of the largest eigenvalue in principal components analysis. *The Annals of Statistics*, 29(2):295–327, 2001.

[16] I.M. Johnstone and A.Y. Lu. On consistency and sparsity for principal components analysis in high dimensions. *Journal of the American Statistical Association*, 104(486):682–693, 2009.

[17] L. Lazzeroni and A. Owen. Plaid models for gene expression data. *Statistica sinica*, 12:61–86, 2002.

[18] Mihee Lee, Haipeng Shen, Jianhua Z. Huang, and J. S. Marron. Biclustering via sparse singular value decomposition. *Biometrics*, 66(4):1087–1095, 2010.

[19] Jinze Liu and Wei Wang. Op-cluster: Clustering by tendency in high dimensional space. In *Proceedings of the Third IEEE International Conference on Data Mining*, ICDM '03, pages 187–, Washington, DC, USA, 2003. IEEE Computer Society.

[20] S.C. Madeira and A.L. Oliveira. Biclustering algorithms for biological data analysis: a survey. *IEEE Transactions on computational Biology and Bioinformatics*, pages 24–45, 2004.

[21] A. Onatski. Asymptotics of the principal components estimator of large factor models with weak factors. *Economics Department, Columbia University*, 2009.

[22] L. Parsons, E. Haque, and H. Liu. Subspace clustering for high dimensional data: a review. *ACM SIGKDD Explorations Newsletter*, 6(1):90–105, 2004.

[23] R.T. Rockafellar. *The theory of subgradients and its applications to problems of optimization. Convex and nonconvex functions.* Heldermann, 1981.

[24] H. Shen and J.Z. Huang. Sparse principal component analysis via regularized low rank matrix approximation. *Journal of multivariate analysis*, 99(6):1015–1034, 2008.

[25] GW Stewart. Perturbation theory for the singular value decomposition. *Computer Science Technical Report Series; Vol. CS-TR-2539*, page 13, 1990.

[26] X. Sun and A. B. Nobel. On the maximal size of Large-Average and ANOVA-fit Submatrices in a Gaussian Random Matrix. *ArXiv e-prints*, September 2010.

[27] A. Tanay, R. Sharan, and R. Shamir. Biclustering algorithms: A survey. *Handbook of computational molecular biology*, 2004.

[28] A.B. Tsybakov. *Introduction to nonparametric estimation.* Springer, 2009.

[29] Lyle Ungar and Dean P. Foster. A formal statistical approach to collaborative filtering. In *CONALD*, 98.

[30] S. Wang, R. R. Gutell, and D. P. Miranker. Biclustering as a method for RNA local multiple sequence alignment. *Bioinformatics*, 23:3289–3296, Dec 2007.

[31] D.M. Witten, R. Tibshirani, and T. Hastie. A penalized matrix decomposition, with applications to sparse principal components and canonical correlation analysis. *Biostatistics*, 10(3):515, 2009.

[32] H. Zou, T. Hastie, and R. Tibshirani. Sparse principal component analysis. *Journal of computational and graphical statistics*, 15(2):265–286, 2006.

